# Support Vector Machines for Multiple-Instance Learning

**Stuart Andrews, Ioannis Tsochantaridis and Thomas Hofmann**
Department of Computer Science, Brown University, Providence, RI 02912
{stu,it,th}@cs.brown.edu

## Abstract

This paper presents two new formulations of multiple-instance learning as a maximum margin problem. The proposed extensions of the Support Vector Machine (SVM) learning approach lead to mixed integer quadratic programs that can be solved heuristically. Our generalization of SVMs makes a state-of-the-art classification technique, including non-linear classification via kernels, available to an area that up to now has been largely dominated by special purpose methods. We present experimental results on a pharmaceutical data set and on applications in automated image indexing and document categorization.

## 1 Introduction

Multiple-instance learning (MIL) [4] is a generalization of supervised classification in which training class labels are associated with sets of patterns, or *bags*, instead of individual patterns. While every pattern may possess an associated true label, it is assumed that pattern labels are only indirectly accessible through labels attached to bags. The law of inheritance is such that a set receives a particular label, if at least one of the patterns in the set possesses the label. In the important case of binary classification, this implies that a bag is "positive" if at least one of its member patterns is a positive example. MIL differs from the general set-learning problem in that the set-level classifier is by design induced by a pattern-level classifier. Hence the key challenge in MIL is to cope with the ambiguity of not knowing which of the patterns in a positive bag are the actual positive examples and which ones are not.

The MIL setting has numerous interesting applications. One prominent application is the classification of molecules in the context of drug design [4]. Here, each molecule is represented by a bag of possible conformations. The efficacy of a molecule can be tested experimentally, but there is no way to control for individual conformations. A second application is in image indexing for content-based image retrieval. Here, an image can be viewed as a bag of local image patches [9] or image regions. Since annotating whole images is far less time consuming then marking relevant image regions, the ability to deal with this type of weakly annotated data is very desirable. Finally, consider the problem of text categorization for which we are the first to apply the MIL setting. Usually, documents which contain a relevant passage are considered to be relevant with respect to a particular cate-

gory or topic, yet class labels are rarely available on the passage level and are most commonly associated with the document as a whole. Formally, all of the above applications share the same type of label ambiguity which in our opinion makes a strong argument in favor of the relevance of the MIL setting.

We present two approaches to modify and extend Support Vector Machines (SVMs) to deal with MIL problems. The first approach explicitly treats the pattern labels as unobserved integer variables, subjected to constraints defined by the (positive) bag labels. The goal then is to maximize the usual pattern margin, or soft-margin, jointly over hidden label variables and a linear (or kernelized) discriminant function. The second approach generalizes the notion of a margin to bags and aims at maximizing the bag margin directly. The latter seems most appropriate in cases where we mainly care about classifying new test bags, while the first approach seems preferable whenever the goal is to derive an accurate pattern-level classifier. In the case of singleton bags, both methods are identical and reduce to the standard soft-margin SVM formulation.

Algorithms for the MIL problem were first presented in [4, 1, 7]. These methods (and related analytical results) are based on hypothesis classes consisting of axis-aligned rectangles. Similarly, methods developed subsequently (e.g., [8, 12]) have focused on specially tailored machine learning algorithms that do not compare favorably in the limiting case of the standard classification setting. A notable exception is [10]. More recently, a kernel-based approach has been suggested which derives MI-kernels on bags from a given kernel defined on the pattern-level [5]. While the MI-kernel approach treats the MIL problem merely as a representational problem, we strongly believe that a deeper conceptual modification of SVMs as outlined in this paper is necessary. However, we share the ultimate goal with [5], which is to make state-of-the-art kernel-based classification methods available for multiple-instance learning.

## 2 Multiple-Instance Learning

In statistical pattern recognition, it is usually assumed that a training set of labeled patterns is available where each pair $(\mathbf{x}_i, y_i) \in \Re^d \times \mathcal{Y}$ has been generated independently from an unknown distribution. The goal is to induce a classifier, i.e., a function from patterns to labels $f : \Re^d \to \mathcal{Y}$. In this paper, we will focus on the binary case of $\mathcal{Y} = \{-1, 1\}$. Multiple-instance learning (MIL) generalizes this problem by making significantly weaker assumptions about the labeling information. Patterns are grouped into *bags* and a label is attached to each bag and not to every pattern. More formally, given is a set of input patterns $\mathbf{x}_1, ..., \mathbf{x}_n$ grouped into bags $\mathbf{B}_1, ..., \mathbf{B}_m$, with $\mathbf{B}_I = \{\mathbf{x}_i : i \in I\}$ for given index sets $I \subseteq \{1, ..., n\}$ (typically non-overlapping). With each bag $\mathbf{B}_I$ is associated a label $Y_I$. These labels are interpreted in the following way: if $Y_I = -1$, then $y_i = -1$ for all $i \in I$, i.e., no pattern in the bag is a positive example. If on the other hand $Y_I = 1$, then at least one pattern $\mathbf{x}_i \in \mathbf{B}_I$ is a positive example of the underlying concept. Notice that the information provided by the label is asymmetric in the sense that a negative bag label induces a unique label for every pattern in a bag, while a positive label does not. In general, the relation between pattern labels $y_i$ and bag labels $Y_I$ can be expressed compactly as $Y_I = \max_{i \in I} y_i$ or alternatively as a set of linear constraints

$$\sum_{i \in I} \frac{y_i + 1}{2} \geq 1, \ \forall I \text{ s.t. } Y_I = 1, \ \text{ and } y_i = -1, \ \forall I \text{ s.t. } Y_I = -1. \tag{1}$$

Finally, let us call a discriminant function $f : \mathcal{X} \to \Re$ *MI-separating* with respect to a multiple-instance data set if $\operatorname{sgn} \max_{i \in I} f(\mathbf{x}_i) = Y_I$ for all bags $\mathbf{B}_I$ holds.

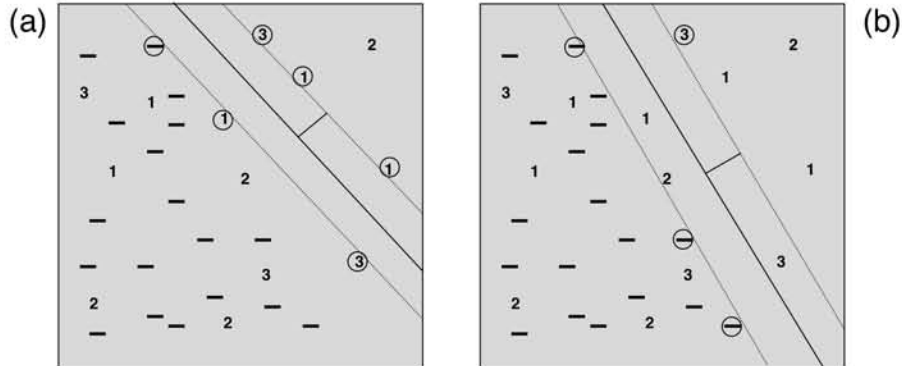

Figure 1: Large margin classifiers for MIL. Negative patterns are denoted by "-" symbols, positive bag patterns by numbers that encode the bag membership. The figure to the left sketches the mi-SVM solution while the figure to the right shows the MI-SVM solution.

## 3 Maximum Pattern Margin Formulation of MIL

We omit an introduction to SVMs and refer the reader to the excellent books on this topic, e.g. [11]. The mixed integer formulation of MIL as a generalized soft-margin SVM can be written as follows in primal form

$$mi\text{-}SVM \qquad \min_{\{y_i\}} \min_{\mathbf{w},b,\xi} \frac{1}{2}\|\mathbf{w}\|^2 + C\sum_i \xi_i \qquad (2)$$

$$\text{s.t.} \qquad \forall i : \ y_i(\langle \mathbf{w}, \mathbf{x}_i \rangle + b) \geq 1 - \xi_i \ , \ \xi_i \geq 0 \ , \ y_i \in \{-1, 1\} \ , \text{and (1) hold.}$$

Notice that in the standard classification setting, the labels $y_i$ of training patterns $\mathbf{x}_i$ would simply be given, while in (2) labels $y_i$ of patterns $\mathbf{x}_i$ not belonging to any negative bag are treated as unknown integer variables. In mi-SVM one thus maximizes a soft-margin criterion jointly over possible label assignments as well as hyperplanes. Figure 1 (a) illustrates this idea for the separable case: We are looking for an MI-separating linear discriminant such that there is at least one pattern from every positive bag in the positive halfspace, while all patterns belonging to negative bags are in the negative halfspace. At the same time, we would like to achieve the maximal margin with respect to the (completed) data set obtained by imputing labels for patterns in positive bags in accordance with Eq. (1).

This is similar to the approach pursued in [6] and [3] for transductive inference. In the latter case, patterns are either labeled or unlabeled. Unlabeled data points are utilized to refine the decision boundary by maximizing the margin on all data points. While the labeling for each unlabeled pattern can be carried out independently in transductive inference, labels of patterns in positive bags are coupled in MIL through the inequality constraints.

The mi-SVM formulation leads to a mixed integer programming problem. One has to find both the optimal labeling and the optimal hyperplane. On a conceptual level this mixed integer formulation captures exactly what MIL is about, i.e. to recover the unobserved pattern labels and to simultaneously find an optimal discriminant. Yet, this poses a computational challenge since the resulting mixed integer programming problem cannot be solved efficiently with state-of-the-art tools, even for moderate size data sets. We will present an optimization heuristic in Section 5.

## 4 Maximum Bag Margin Formulation of MIL

An alternative way of applying maximum margin ideas to the MIL setting is to extend the notion of a margin from individual patterns to sets of patterns. It is natural to define the functional margin of a bag with respect to a hyperplane by

$$\gamma_I \equiv Y_I \max_{i \in I}(\langle \mathbf{w}, \mathbf{x}_i \rangle + b). \tag{3}$$

This generalization reflects the fact that predictions for bag labels take the form $\hat{Y}_I = \operatorname{sgn} \max_{i \in I}(\langle \mathbf{w}, \mathbf{x}_i \rangle + b)$. Notice that for a positive bag the margin is defined by the margin of the "most positive" pattern, while the margin of a negative bag is defined by the "least negative" pattern. The difference between the two formulations of maximum-margin problems is illustrated in Figure 1. For the pattern-centered mi-SVM formulation, the margin of every pattern in a positive bag matters, although one has the freedom to set their label variables so as to maximize the margin. In the bag-centered formulation, only one pattern per positive bag matters, since it will determine the margin of the bag. Once these "witness" patterns have been identified, the relative position of other patterns in positive bags with respect to the classification boundary becomes irrelevant. Using the above notion of a *bag margin*, we define an MIL version of the soft-margin classifier by

$$\text{MI-SVM} \qquad \min_{\mathbf{w}, b, \xi} \frac{1}{2}\|\mathbf{w}\|^2 + C \sum_I \xi_I \tag{4}$$

$$\text{s.t.} \quad \forall I : Y_I \max_{i \in I}(\langle \mathbf{w}, \mathbf{x}_i \rangle + b) \geq 1 - \xi_I, \ \xi_I \geq 0.$$

For negative bags one can unfold the max operation by introducing one inequality constraint per pattern, yet with a single slack variable $\xi_I$. Hence the constraints on negative bag patterns, where $Y_I = -1$, read as $-\langle \mathbf{w}, \mathbf{x}_i \rangle - b \geq 1 - \xi_I, \ \forall i \in I$.

For positive bags, we introduce a selector variable $s(I) \in I$ which denotes the pattern selected as the positive "witness" in $\mathbf{B}_I$. This will result in constraints $\langle \mathbf{w}, \mathbf{x}_{s(I)} \rangle + b \geq 1 - \xi_I$. Thus we arrive at the following equivalent formulation

$$\min_{s} \min_{\mathbf{w}, b, \xi} \frac{1}{2}\|\mathbf{w}\|^2 + C \sum_I \xi_I \tag{5}$$

$$\text{s.t.} \quad \forall I : Y_I = -1 \ \wedge \ -\langle \mathbf{w}, \mathbf{x}_i \rangle - b \geq 1 - \xi_I, \ \forall i \in I,$$

$$\text{or} \quad Y_I = 1 \quad \wedge \ \langle \mathbf{w}, \mathbf{x}_{s(I)} \rangle + b \geq 1 - \xi_I, \ \text{and} \ \xi_I \geq 0. \tag{6}$$

In this formulation, every positive bag $\mathbf{B}_I$ is thus effectively represented by a single member pattern $\mathbf{x}_I \equiv \mathbf{x}_{s(I)}$. Notice that "non-witness" patterns ($\mathbf{x}_i$, $i \in I$ with $i \neq s(I)$) have no impact on the objective.

For given selector variables, it is straightforward to derive the dual objective function which is very similar to the standard SVM Wolfe dual. The only major difference is that the box constraints for the Lagrange parameters $\alpha$ are modified compared to the standard SVM solution, namely one gets

$$0 \leq \alpha_I \leq C, \quad \text{for } I \text{ s.t. } Y_I = 1 \quad \text{and} \quad 0 \leq \sum_{i \in I} \alpha_i \leq C, \quad \text{for } I \text{ s.t. } Y_I = -1. \tag{7}$$

Hence, the influence of each *bag* is bounded by $C$.

## 5 Optimization Heuristics

As we have shown, both formulations, mi-SVM and MI-SVM, can be cast as mixed-integer programs. In deriving optimization heuristics, we exploit the fact that for

```
initialize y_i = Y_I for i ∈ I
REPEAT
    compute SVM solution w,b for data set with imputed labels
    compute outputs f_i = ⟨w,x_i⟩ + b for all x_i in positive bags
    set y_i = sgn(f_i) for every i ∈ I, Y_I = 1
    FOR (every positive bag B_I)
        IF (∑_{i∈I}(1 + y_i)/2 == 0)
            compute i* = arg max_{i∈I} f_i
            set y_{i*} = 1
        END
    END
WHILE (imputed labels have changed)
OUTPUT (w,b)
```

Figure 2: Pseudo-code for mi-SVM optimization heuristics (synchronous update).

```
initialize x_I = ∑_{i∈I} x_i/|I| for every positive bag B_I
REPEAT
    compute QP solution w,b for data set with
        positive examples {x_I : Y_I = 1}
    compute outputs f_i = ⟨w,x_i⟩ + b for all x_i in positive bags
    set x_I = x_{s(I)}, s(I) = arg max_{i∈I} f_i for every I, Y_I = 1
WHILE (selector variables s(I) have changed)
OUTPUT (w,b)
```

Figure 3: Pseudo-code for MI-SVM optimization heuristics (synchronous update).

given integer variables, i.e. the hidden labels in mi-SVM and the selector variables in MI-SVM, the problem reduces to a QP that can be solved exactly. Of course, all the derivations also hold for general kernel functions $K$.

A general scheme for a simple optimization heuristic may be described as follows. Alternate the following two steps: (i) for given integer variables, solve the associated QP and find the optimal discriminant function, (ii) for a given discriminant, update one, several, or all integer variables in a way that (locally) minimizes the objective. The latter step may involve the update of a label variable $y_i$ of a single pattern in mi-SVM, the update of a single selector variable $s(I)$ in MI-SVM, or the simultaneous update of all integer variables. Since the integer variables are essentially decoupled given the discriminant (with the exception of the bag constraints in mi-SVM), this can be done very efficiently. Also notice that we can re-initialize the QP-solver at every iteration with the previously found solution, which will usually result in a significant speed-up. In terms of initialization of the optimization procedure, we suggest to impute positive labels for patterns in positive bags as the initial configuration in mi-SVM. In MI-SVM, $x_I$ is initialized as the centroid of the bag patterns. Figure 2 and 3 summarize pseudo-code descriptions for the algorithms utilized in the experiments.

There are many possibilities to refine the above heuristic strategy, for example, by starting from different initial conditions, by using branch and bound techniques to explore larger parts of the discrete part of the search space, by performing stochastic updates (simulated annealing) or by maintaining probabilities on the integer variables in the spirit of deterministic annealing. However, we have been able to achieve competitive results even with the simpler optimization heuristics, which val-

|  | EMDD[12] | DD [9] | MI-NN [10] | IAPR [4] | mi-SVM | MI-SVM |
|---|---|---|---|---|---|---|
| MUSK1 | 84.8 | 88.0 | 88.9 | **92.4** | 87.4 | 77.9 |
| MUSK2 | 84.9 | 84.0 | 82.5 | **89.2** | 83.6 | 84.3 |

Table 1: Accuracy results for various methods on the MUSK data sets.

idate the maximum margin formulation of SVM. We will address further algorithmic improvements in future work.

# 6  Experimental Results

We have performed experiments on various data sets to evaluate the proposed techniques and compare them to other methods for MIL. As a reference method we have implemented the EM Diverse Density (EM-DD) method [12], for which very competitive results have been reported on the MUSK benchmark[1].

## 6.1  MUSK Data Set

The MUSK data sets are *the* benchmark data sets used in virtually all previous approaches and have been described in detail in the landmark paper [4]. Both data sets, MUSK1 and MUSK2, consist of descriptions of molecules using multiple low-energy conformations. Each conformation is represented by a 166-dimensional feature vector derived from surface properties. MUSK1 contains on average approximately 6 conformation per molecule, while MUSK2 has on average more than 60 conformations in each bag. The averaged results of ten 10-fold cross-validation runs are summarized in Table 1. The SVM results are based on an RBF kernel $K(\mathbf{x}, \mathbf{y}) = \exp(-\gamma \|\mathbf{x} - \mathbf{y}\|^2)$ with coarsely optimized $\gamma$. For both MUSK1 and MUSK2 data sets, mi-SVM achieves competitive accuracy values. While MI-SVM outperforms mi-SVM on MUSK2, it is significantly worse on MUSK1. Although both methods fail to achieve the performance of the best method (iterative APR)[2], they compare favorably with other approaches to MIL.

## 6.2  Automatic Image Annotation

We have generated new MIL data sets for an image annotation task. The original data are color images from the Corel data set that have been preprocessed and segmented with the Blobworld system [2]. In this representation, an image consists of a set of segments (or blobs), each characterized by color, texture and shape descriptors. We have utilized three different categories ("elephant", "fox", "tiger") in our experiments. In each case, the data sets have 100 positive and 100 negative example images. The latter have been randomly drawn from a pool of photos of other animals. Due to the limited accuracy of the image segmentation, the relative small number of region descriptors and the small training set size, this ends up being quite a hard classification problem. We are currently investigating alternative image

| Data Set | Dims | EM-DD | mi-SVM | | | MI-SVM | | |
|----------|------|-------|--------|---|---|--------|---|---|
| Category | inst/feat | | linear | poly | rbf | linear | poly | rbf |
| Elephant | 1391/230 | 78.3 | **82.2** | 78.1 | 80.0 | 81.4 | 79.0 | 73.1 |
| Fox | 1320/230 | 56.1 | 58.2 | 55.2 | 57.9 | 57.8 | **59.4** | 58.8 |
| Tiger | 1220/230 | 72.1 | 78.4 | 78.1 | 78.9 | **84.0** | 81.6 | 66.6 |

Table 2: Classification accuracy of different methods on the Corel image data sets.

| Data Set | Dims | EM-DD | mi-SVM | | | MI-SVM | | |
|----------|------|-------|--------|---|---|--------|---|---|
| Category | inst/feat | | linear | poly | rbf | linear | poly | rbf |
| TST1 | 3224/6668 | 85.8 | 93.6 | 92.5 | 90.4 | **93.9** | 93.8 | 93.7 |
| TST2 | 3344/6842 | 84.0 | 78.2 | 75.9 | 74.3 | **84.5** | 84.4 | 76.4 |
| TST3 | 3246/6568 | 69.0 | **87.0** | 83.3 | 69.0 | 82.2 | 85.1 | 77.4 |
| TST4 | 3391/6626 | 80.5 | 82.8 | 80.0 | 69.6 | 82.4 | **82.9** | 77.3 |
| TST7 | 3367/7037 | 75.4 | **81.3** | 78.7 | 81.3 | 78.0 | 78.7 | 64.5 |
| TST9 | 3300/6982 | 65.5 | **67.5** | 65.6 | 55.2 | 60.2 | 63.7 | 57.0 |
| TST10 | 3453/7073 | 78.5 | 79.6 | 78.3 | 52.6 | 79.5 | **81.0** | 69.1 |

Table 3: Classification accuracy of different methods on the TREC9 document categorization sets.

representations in the context of applying MIL to content-based image retrieval and automated image indexing, for which we hope to achieve better (absolute) classification accuracies. However, these data sets seem legitimate for a comparative performance analysis. The results are summarized in Table 2. They show that both, mi-SVM and MI-SVM achieve a similar accuracy and outperform EM-DD by a few percent. While MI-SVM performed marginally better than mi-SVM, both heuristic methods were susceptible to other nearby local minima. Evidence of this effect was observed through experimentation with asynchronus updates, as described in Section 5, where we varied the number of integer variables updated at each iteration.

## 6.3  Text Categorization

Finally, we have generated MIL data sets for text categorization. Starting from the publicly available TREC9 data set, also known as OHSUMED, we have split documents into passages using overlapping windows of maximal 50 words each. The original data set consists of several years of selected MEDLINE articles. We have worked with the 1987 data set used as training data in the TREC9 filtering task which consists of approximately 54,000 documents. MEDLINE documents are annotated with MeSH terms (Medical Subject Headings), each defining a binary concept. The total number of MeSH terms in TREC9 was 4903. While we are currently performing a larger scale evaluation of MIL techniques on the full data set, we report preliminary results here on a smaller, randomly subsampled data set. We have been using the first seven categories of the pre-test portion with at least 100 positive examples. Compared to the other data sets the representation is extremely sparse and high-dimensional, which makes this data an interesting additional benchmark. Again, using linear and polynomial kernel functions, which are generally known to work well for text categorization, both methods show improved performance over EM-DD in almost all cases. No significant difference between the two methods is clearly evident for the text classification task.

# 7 Conclusion and Future Work

We have presented a novel approach to multiple-instance learning based on two alternative generalizations of the maximum margin idea used in SVM classification. Although these formulations lead to hard mixed integer problems, even simple local optimization heuristics already yield quite competitive results compared to the baseline approach. We conjecture that better optimization techniques, that can for example avoid unfavorable local minima, may further improve the classification accuracy. Ongoing work will also extend the experimental evaluation to include larger scale problems.

As far as the MIL research problem is concerned, we have considered a wider range of data sets and applications than is usually done and have been able to obtain very good results across a variety of data sets. We strongly suspect that many MIL methods have been optimized to perform well on the MUSK benchmark and we plan to make the data sets used in the experiments available to the public to encourage further empirical comparisons.

**Acknowledgments**

This work was sponsored by an NSF-ITR grant, award number IIS-0085836.

## Footnotes

[1]However, the description of EM-DD in [12] seems to indicate that the authors used the test data to select the optimal solution obtained from multiple runs of the algorithm. In the pseudo-code formulation of EM-DD, $D_i$ is used to compute the error for the $i$-th data fold, where it should in fact be $D_t = D - D_i$ (using the notation of [12]). We have used the corrected version of the algorithm in our experiments and have obtained accuracy numbers using EM-DD that are more in line with previously published results.

[2]Since the IAPR (iterative axis parallel rectangle) methods in [4] have been specifically designed and optimized for the MUSK classification task, the superiority of APR should not be interpreted as a failure.

# References

[1] P. Auer. On learning from multi-instance examples: Empirical evaluation of a theoretical approach. In *Proc. 14th International Conf. on Machine Learning*, pages 21–29. Morgan Kaufmann, San Francisco, CA, 1997.

[2] C. Carson, M. Thomas, S. Belongie, J. M. Hellerstein, and J. Malik. Blobworld: A system for region-based image indexing and retrieval. In *Proceedings Third International Conference on Visual Information Systems*. Springer, 1999.

[3] A. Demirez and K. Bennett. Optimization approaches to semisupervised learning. In M. Ferris, O. Mangasarian, and J. Pang, editors, *Applications and Algorithms of Complementarity*. Kluwer Academic Publishers, Boston, 2000.

[4] T. G. Dietterich, R. H. Lathrop, and T. Lozano-Perez. Solving the multiple instance problem with axis-parallel rectangles. *Artificial Intelligence*, 89(1-2):31–71, 1997.

[5] T. Gärtner, P. A. Flach, A. Kowalczyk, and A. J. Smola. Multi-instance kernels. In *Proc. 19th International Conf. on Machine Learning*. Morgan Kaufmann, San Francisco, CA, 2002.

[6] T. Joachims. Transductive inference for text classification using support vector machines. In *Proceedings 16th International Conference on Machine Learning*, pages 200–209. Morgan Kaufmann, San Francisco, CA, 1999.

[7] P.M. Long and L. Tan. PAC learning axis aligned rectangles with respect to product distributions from multiple-instance examples. In *Proc. Comp. Learning Theory*, 1996.

[8] O. Maron and T. Lozano-Pérez. A framework for multiple-instance learning. In *Advances in Neural Information Processing Systems*, volume 10. MIT Press, 1998.

[9] O. Maron and A. L. Ratan. Multiple-instance learning for natural scene classification. In *Proc. 15th International Conf. on Machine Learning*, pages 341–349. Morgan Kaufmann, San Francisco, CA, 1998.

[10] J. Ramon and L. De Raedt. Multi instance neural networks. In *Proceedings of ICML-2000, Workshop on Attribute-Value and Relational Learning*, 2000.

[11] B. Schölkopf and A. Smola. *Learning with Kernels. Support Vector Machines, Regularization, Optimization and Beyond*. MIT Press, 2002.

[12] Qi Zhang and Sally A. Goldman. EM-DD: An improved multiple-instance learning technique. In *Advances in Neural Information Processing Systems*, volume 14. MIT Press, 2002.